# Randomized Algorithms for Comparison-based Search

**Dominique Tschopp**
AWK Group
Bern, Switzerland
dominique.tschopp@gmail.com

**Suhas Diggavi**
University of California Los Angeles (UCLA)
Los Angeles, CA 90095
suhasdiggavi@ucla.edu

**Payam Delgosha**
Sharif University of Technology
Tehran, Iran
pdelgosha@ee.sharif.ir

**Soheil Mohajer**
Princeton University
Princeton, NJ 08544
smohajer@princeton.edu

## Abstract

This paper addresses the problem of finding the nearest neighbor (or one of the $R$-nearest neighbors) of a query object $q$ in a database of $n$ objects, when we can only use a comparison oracle. The comparison oracle, given two reference objects and a query object, returns the reference object most similar to the query object. The main problem we study is how to search the database for the nearest neighbor (NN) of a query, while minimizing the questions. The difficulty of this problem depends on properties of the underlying database. We show the importance of a characterization: *combinatorial disorder* $D$ which defines approximate triangle inequalities on ranks. We present a lower bound of $\Omega(D \log \frac{n}{D} + D^2)$ average number of questions in the search phase for any randomized algorithm, which demonstrates the fundamental role of $D$ for worst case behavior. We develop a randomized scheme for NN retrieval in $O(D^3 \log^2 n + D \log^2 n \log \log n^{D^3})$ questions. The learning requires asking $O(n D^3 \log^2 n + D \log^2 n \log \log n^{D^3})$ questions and $O(n \log^2 n / \log(2D))$ bits to store.

## 1 Introduction

Consider the situation where we want to search and navigate a database, but the underlying relationships between the objects are unknown and are accessible only through a comparison oracle. The comparison oracle, given two reference objects and a query object, returns the reference object most similar to the query object. Such an oracle attempts to model the behavior of human users, capable of making statements about similarity, but not of assigning meaningful numerical values to distances between objects. These situations could occur in many tasks, such as recommendation for movies, restaurants etc., or a human-assisted search system for image databases among other applications. Using such an oracle, the best we can hope for is to obtain, for every object $u$ in the database, a ranking of the other objects according to their similarity to $u$. However, the use of the oracle to get complete information about ranking could be costly, since invoking the oracle is to represent human input to the task (preferences in movies, comparison of images etc). We can pre-process the database by asking questions during a learning phase, and use the resulting answers to facilitate the search process. Therefore, the main question we ask in this paper is to design a (approximate) nearest neighbor retrieval algorithm while minimizing the number of questions to such an oracle.

Clearly the difficulty of searching using such an oracle depends critically on the properties of the set of objects. We demonstrate the importance of a characterization which determines the performance

of comparison based search algorithms. *Combinatorial disorder* (introduced by Goyal et al. [1]), defines approximate triangle inequalities on ranks. Roughly speaking, it defines a multiplicative factor $D$ by which the triangle inequality on ranks can be violated. We show our first lower bound of $\Omega(D \log \frac{n}{D} + D^2)$ on average number of questions in the search phase for any randomized algorithm, and therefore demonstrate the fundamental importance of $D$ for worst case behavior. When the disorder is known, we can use partial rank information to estimate, or infer the other ranks. This allows us to design a novel hierarchical scheme which considerably improves the existing bounds for nearest neighbor search based on a similarity oracle, and performs provably close to the lower bound. If no characterization of the hidden space can be used as an input, we develop algorithms that can decompose the space such that dissimilar objects are likely to get separated, and similar objects have the tendency to stay together; generalizing the notion of randomized $k$-$d$-trees [2]. This is developed in more detail in [3]. Due to space constraints, we give statements of the results along with an outline of proof ideas in the main text. Additionally we provide proof details in the appendix [4] as extra material allowed by NIPS.

**Relationship to published works:** Nearest neighbor (NN) search problem has been very well studied for metric spaces (see [5]). However, in all these works, it is assumed that one can compute distances between points in the data set. In [6, 7, 8, 9, 10, 11], various approaches to measure similarities between images are presented, which could be used as comparison oracles in our setup. The algorithmic aspects of searching with a comparison oracle was first studied in [1], where a random walk algorithm is presented. The main limitation of this algorithm is the fact that all rank relationships need to be known in advance, which amounts to asking the oracle $O(n^2 \log n)$ questions, in a database of size $n$. In [12], a data structure similar in spirit to $\epsilon$-nets of [13] is introduced. It is shown that a learning phase with complexity $O(D^7 n \log^2 n)$ questions and a space complexity of $O(D^5 n + Dn \log n)$ allows to retrieve the NN in $O(D^4 \log n)$ questions. The learning phase builds a hierarchical structure based on coverings of exponentially decreasing radii. In this paper, we present what we believe is the first *lower bound* for search through comparisons. This gives a more fundamental meaning to $D$ as a parameter determining worst case behavior. Based on the insights gained from this worst case analysis, we then improve (see Section 3) the existing upper bounds by a $\mathrm{poly}(D)$ factor, if we are willing to accept a negligible (less than $\frac{1}{n}$) probability of failure. Our algorithm is based on random sampling, and can be seen as a form of metric skip list (as introduced in [14]), but applied to a combinatorial (non-metric) framework. However, the fact that we do not have access to distances forces us to use new techniques in order to minimize the number of questions (or ranks we need to compute). In particular, we sample the database at different densities, and infer the ranks from the density of the sampling, which we believe is a new technique. We also need to relate samples to each other when building the data structure top down.

A natural question to ask is whether one can develop data structures for NN when a characterization of the underlying space is unknown. In [2], when one has access to metric distances, a binary tree decomposition of a dataset that adapts to its "intrinsic dimension" [13] has been designed. We extend the result of [2] to our setup, where we have a comparison oracle but do not have access to metric distances. This can be used in a manner similar to [2] to find (approximate) NN (see [3] for more details).

To the best of our knowledge, the notion of randomized NN search using similarity oracle is studied for the first time in this paper. Moreover, the hierarchical search scheme proposed is more efficient than earlier schemes. The lower bound presented appears to be new and demonstrates that our schemes are (almost) efficient.

## 2 Definitions and Problem Statement

We consider a hidden space $\mathcal{K}$, and a database of objects $\mathcal{T} \subset \mathcal{K}$, with $|\mathcal{T}| = n$. We can only access this space through a *similarity oracle* which for any point $q \in \mathcal{K}$, and objects $u, v \in \mathcal{T}$ returns

$$\mathcal{O}(q, u, v) = \begin{cases} u & \text{if } u \text{ is more similar to } q \text{ than } v \\ v & \text{else.} \end{cases} \tag{1}$$

The goal is to develop and analyse algorithms which for any given $q \in \mathcal{K}$, can find an object in the database $a \in \mathcal{T}$ which is the nearest neighbor (NN) to $q$, using the smallest number of questions of type (1). We also relax this goal to find the approximate NN with "high probability". The algorithm

may have a learning phase, in which it explores the structure of the database, and stores it using a certain amount of memory. Note that this phase has to be done prior to knowing the query $q \in \mathcal{K}$. Then, once the query is given, the search phase of the algorithm asks a certain number of questions of type (1) and finds the closest object in the database.

The performance of the algorithm is measured by three components among which there could be a trade-off: the number of questions asked in the learning phase, the number of questions asked in the searching phase, and the total memory to be stored. The main goal of this work is to design algorithms for NN search and characterize its performance in terms of these parameters. We will present some definitions which are required to state the results of this paper.

**Definition 1.** *The rank of $u$ in a set $\mathcal{S}$ with respect to $v$, $r_v(u, \mathcal{S})$ is equal to c, if $u$ is the $c^{th}$ nearest object to $v$ in $\mathcal{S}$, i.e., $|\{w \in \mathcal{S} : d(w,v) < d(u,v)\}| = c - 1$, where $d(w,v) < d(u,v)$ could be interpreted as a distance function. Also the rank ball $\beta_x(r)$ is defined to be $\{y : r_x(y, \mathcal{S})) \leq r\}$.*

Note that we do not need existence of a distance function in Definition 1. We could replace $d(w,v) < d(u,v)$ with "$v$ is more similar to $w$ than $u$" by using the oracle in (1).

To simplify the notation, we only indicate the set if it is unclear from the context *i.e.,* we write $r_v(u)$ instead of $r_v(u, \mathcal{S})$ unless there is an ambiguity. Note that rank need not be a symmetric relationship between objects *i.e.,* $r_u(v) \neq r_v(u)$ in general. Further, note that we can rank $m$ objects w.r.t. an object $o$ by asking the oracle $O(m \log m)$ questions, using standard sort algorithms [15].

Our characterization of the space of objects is through a form of approximate triangle inequalities introduced in [1] and [12]. Instead of defining a inequalities between distances, these triangle inequalities defined over ranks, and depend on a property of the space, called the *disorder constant*.

**Definition 2.** *The combinatorial disorder of a set of objects S is the smallest D such that $\forall x, y, z \in S$, we have the following approximate triangle inequalities:*

*(i)* $r_x(y, S) \leq D(r_z(x, S) + r_z(y, S))$  　　　 *(ii)* $r_x(y, S) \leq D(r_x(z, S) + r_y(z, S))$

*(iii)* $r_x(y, S) \leq D(r_x(z, S) + r_z(y, S))$  　　 *(iv)* $r_x(y, S) \leq D(r_z(x, S) + r_y(z, S))$

*In particular, $r_x(x, S) = 0$ and $r_x(y, S) \leq Dr_y(x, S)$.*

## 3   Contributions

Our contributions are the following: **(i)** we design a randomized hierarchical data structure with which we can do NN search using the comparison oracle **(ii)** we develop the first lower bound for the search complexity in the combinatorial framework of [1, 12], and thereby demonstrate the importance of combinatorial disorder. The performance of the randomized algorithm (see **(i)**) is shown to be close to this lower bound. We also develop a binary tree decomposition that adapts to the data set in a manner analogous to [2].

More precisely, we prove a lower bound on the average search time to retrieve the nearest neighbor of a query point for randomized algorithms in the combinatorial framework.

**Theorem 1.** *There exists a space, a configuration of a database of $n$ objects in that space that for the uniform distribution over placements of the query point $q$ such that no randomized search algorithm, even if $O(n^3)$ questions can be asked in the learning phase, can find $q$'s nearest neighbor in the database for sure (with a probability of error of 0) by asking less than an expected $\Omega(D^2 + D \log n/D)$ questions in the worst case when $D < \sqrt{n}$.*

As a consequence of this theorem, there must exist at least one query point in this configuration which requires asking at least $\Omega(D \log(\frac{n}{D}) + D^2)$ questions, hence setting a lower bound on the search complexity. Based on the insights gained from this worst case analysis, we introduce a conceptually simple randomized hierarchical scheme that allows us to reduce the learning compared to the existing algorithm (see [12, 1]) by a factor $D^4$, memory consumption by a factor $D^5/\log^2 n$, and a factor $D/\log n \log \log n^{D^3}$ for search.

**Theorem 2.** *We design a randomized algorithm, which for a given query point $q$, can retrieve its nearest neighbor with high probability in $O(D^3 \log^2 n + D \log^2 n \log \log n^{D^3})$ questions. The*

*learning requires asking $O(nD^3 \log^2 n + D \log^2 n \log\log n^{D^3})$ questions and we need to store $O(n \log^2 n / \log(2D))$ bits.*

Consequently, our schemes are asymptotically (for $n$) within $D\mathrm{polylog}(n)$ questions of the optimal search algorithm.

## 4 Lower Bounds for NNS

A natural question to ask is whether there are databases and query points for which we need to ask a minimal number of questions, independent of the algorithm used. In this section, we construct a database $\mathcal{T}$ of $n$ objects, a universe of queries $\mathcal{K}\backslash\mathcal{T}$ and similarity relationships, for which no search algorithm can find the NN of a query point in less than expected $\Omega(D \log \frac{n}{D} + D^2)$ questions. We show this even when all possible questions $\mathcal{O}(u, v, w)$ related to the $n$ database objects (*i.e.,* $u, v, w \in \mathcal{T}$) can be asked during the learning phase. The query is chosen uniformly from the universe of queries and is unknown during the learning phase.

**Database Structure:** Consider the weighted graph shown in Fig. 1. It consists of a star with $\alpha$ branches $\phi_1, \phi_2, \ldots, \phi_\alpha$, each composed of $n/\alpha^2$ supernodes (SN). Each of the supernodes in turn contains $\alpha$ database objects (i.e., objects in $\mathcal{T}$). Clearly, in total there are $\alpha\alpha\frac{n}{\alpha^2} = n$ objects. Note that the database $\mathcal{T}$ only includes the set of objects inside the supernodes, and the supernodes, themselves, are *not* element of $\mathcal{T}$. We indicate the objects in each branch by numbers from $1$ to $n/\alpha$.

We define the set of queries, $\mathcal{M}$, as follows: every query point $q$ is attached to one object form $\mathcal{T}$ on each branch of the star with an edge; this object is called a *direct node* (DN) on the corresponding branch. Moreover, we assume that the weights of all *query edges*, the $\alpha$ edges connecting the query to its DNs, are different. Therefore, the set of all queries, $\mathcal{M}$ could be restricted to $\alpha!(n/\alpha)^\alpha$ elements, since there are $n/\alpha$ choices for choosing the direct node in each branch (*i.e.,* $(n/\alpha)^\alpha$ choices for $\alpha$ branches), and the weight of the query edges can be ordered in $\alpha!$ different ways.

In this example, distance between two nodes is given by the weighted graph distance, and the oracle answers queries based on this distance. All edges connecting the SNs to each other have weight 1 expect those $\alpha$ edges emitting from the center of the star and ending at the first SNs which have weight $n/(\alpha^2)$. Edges connecting the objects in a supernode to its root are called *object* edges. We assume that all $n/\alpha$ object edges in branch $\phi_i$ have weight $i/(4\alpha)$. It remains to fix the weight of the query edges. We will define the weight of these edges in the following.

**Definition 3.** *For a query $q \in \mathcal{M}$, define the $\alpha$-tuple $\delta_q \in \{1, 2, \ldots, n/\alpha\}^\alpha$ to be the sequence of DNs of $q$ in $\alpha$ branches,* i.e., *$\delta_q(i)$ denotes the indicator of the object on $\phi_i$ which is connected to $q$ via a query edge. We also represent the rank of the DNs w.r.t. $q$, by an $\alpha$-tuple $\Psi_q \in \{1, \ldots, \alpha\}^\alpha$,* i.e., *$\Psi_q(i)$ denotes the rank of the DN on branch $\phi_i$ among all the other DNs w.r.t. $q$.*

Now we can define the weight of query edges. For a query $q \in \mathcal{M}$, the weight of the query edge which connects $q$ to $\delta_q(i)$ is given to be $1 + (\Psi_q(i)/\alpha)\epsilon$, where $\epsilon \ll 1/(4\alpha)$ is a constant.

As mentioned before, the disorder constant plays an important role in the performance of the algorithm. The following lemma gives the disorder constant for the database introduced. The proof of this lemma is presented in the appendix [4].

**Lemma 1.** *The star shaped graph introduced above has disorder constant $D = \Theta(\alpha)$.*

**The Lower Bound:** In the proof of Theorem 1, we will use Yao's minimax principle (see [16]), which states that, for any distribution on the inputs the expected cost for the best deterministic algorithm provides a lower bound on the worst case running time of any randomized algorithm. In the following, we state two lower bounds for the number of questions in the searching phase of any deterministic algorithm for the database illustrated in Fig. 1.

**Proposition 1.** *The number of questions asked by a deterministic algorithm $\mathcal{A}$, on average w.r.t. uniform distribution, to solve the NNS problem in star graph, is lower bounded by $\Omega\left(\alpha \log(n/\alpha)\right)$.*

To outline the proof of this claim: each question asked by the algorithm involves two database nodes. Note that the weights of the edges emitting from the center of the graph are chosen so that the branches become *independent*, in the sense that questioning nodes on one branch will not reveal

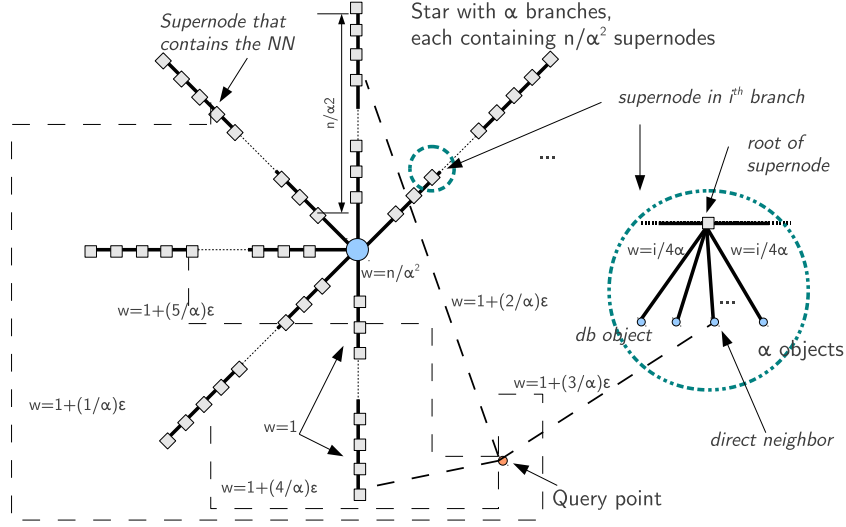

Figure 1: The star database: a weighted star graph with $\alpha$ branches, each composed of $n/\alpha^2$ "supernodes". Each supernode further includes $\alpha$ database objects. Finally, each query points is randomly connected to *one* object on *each* branch of the star via a weighted edge. The weights of the edges are chosen so than the disorder constant be $D = \Theta(\alpha)$.

any information about other branches. Therefore, in order to find the nearest node to $q$, the algorithm has to find the direct node on each branch, and then compare them to find the NN. For any branch $\phi_i$, there are $n/\alpha$ candidates which can be DN of $q$ with equal probability. Hence, roughly speaking, the algorithm needs to ask $\Omega(\log(n/\alpha))$ questions for each branch. This yields to a minimum total of $\Omega(\alpha \log(n/\alpha))$ questions for $\alpha$ *independent* branches in the graph.

**Proposition 2.** *Any deterministic algorithm $\mathcal{A}$ solving nearest neighbor search problem in the input query set $\mathcal{M}$ with uniform distribution should ask on average $\Omega(\alpha^2)$ questions from the oracle.*

To outline the proof of this claim: consider an arbitrary branch $\phi_i$ and assume a genie tells us that which supernode on $\phi_i$ contains the DN for $q$. However, we do not know which of $p_1, p_2, \ldots, p_\alpha$, the nodes inside the revealed supernode, is the DN of $q$ on $\phi_i$. Since all the edges connecting the supernode to its children have the same weight, questioning just some of them is not sufficient to find the direct node, and effectively all of them should be asked on average. Since each question involves at most two of such nodes, an $\Omega(\alpha)$ questions is required to find the DN on $\phi_i$. Summing up the same number over all $\alpha$ branches, we obtain the $\Omega(\alpha^2)$ lower bound on the number of questions.

Theorem 1 is a direct consequence of the above mentioned propositions.

*Proof of Theorem 1.* Let $\mathcal{A}$ be an arbitrary deterministic algorithm which solves NNS problem in star shaped graph with uniform distribution. If $Q_\mathcal{A}$ denotes the average number of questions $\mathcal{A}$ asks, according to Proposition 1 and Proposition 2 we have

$$Q_\mathcal{A} \geq \max \left\{ \Omega\left(\alpha \log \frac{n}{\alpha}\right), \Omega(\alpha^2) \right\} \geq \frac{1}{2}\left(\Omega\left(\alpha \log \frac{n}{\alpha}\right), \Omega(\alpha^2)\right) = \Omega\left(\alpha^2 + \alpha \log n/\alpha\right). \quad (2)$$

By using the Yao's Minimax principle, we can conclude Theorem 1. $\qquad\square$

We can show that this bound is best bound one can find for this dataset. Indeed, we present an algorithm in the appendix [4], which finds the query by asking $\Theta\left(\alpha^2 + \alpha \log n/\alpha\right)$ questions.

# 5 Hierarchical Data Structure For Nearest-Neighbor Search

In this section we develop the search algorithm that guarantees the performance stated in Theorem 2. The learning phase is described in Algorithm 1. The algorithm builds a hierarchical decomposition level by level, top-down. At each level, we sample objects from the database. The set of samples at level $i$ is denoted by $S_i$, and we have $|S_i| = m_i = a(2D)^i \log n$, where $a$ is a constant independent[1] of $n$ and $D$. At each level $i$, every object in $\mathcal{T}$ is put in the "bin" of the sample in $S_i$ closest to it. To find this sample at level $i$, for every object $p$ we rank the samples in $S_i$ w.r.t. $p$ (by using the oracle to make pairwise comparisons). However, we show that given that we know $D$, we only need to rank those samples that fell in the bin of one of the at most $4aD \log n$ nearest samples to $p$ at level $i - 1$. This is a consequence of the fact that we carefully chose the density of the samples at each level. Further, the fact that we build the hierarchy top-down, allows us to use the answers to the questions asked at level $i$, to reduce the number of questions we need to ask at level $i + 1$. This way, the number of questions per object does not increase as we go down in the hierarchy, even though the number of samples increases.

For object $p$, $\nu_p(i)$ denotes the nearest neighbor to object $p$ in $S_i$. We want to keep the $\lambda_i = n/(2D)^{i-1}$ closest objects in $S_i$ to $p$ in the set $\Gamma_p(i)$, *i.e.*, all objects $o \in S_i$ so that $r_p(o, S_i) \le \lambda_i$. It could be shown that for an object $o$ to be in $\Gamma_p(i)$ it is necessary that $\nu_o(i - 1)$ be in $\Gamma_p(i - 1)$. Therefore by taking $\Lambda_p(i) = \{o \in S_i | \nu_o(i - 1) \in \Gamma_p(i - 1)\}$ we have $\Gamma_p(i) \subseteq \Lambda_p(i)$. It could be verified that $|\Gamma_p(i)| \le 4aD \log n$, therefore $\Gamma_p(i)$ can be constructed by finding the $4aD \log n$ closest objects in $\Lambda_p(i)$ to $p$. Definitely the first object in $\Gamma_p(i)$ is $\nu_p(i)$. Therefore we can recursively build $\Gamma_p(i)$, $\Lambda_p(i)$ and $\nu_p(i)$ for $1 \le i \le \log n / \log 2D$ for any object $p$, as it is done in the algorithm.

The role of macros `BuildHeap` and `ExtractMin` is to build a heap from unordered data, and extract the minimum element from the heap, respectively. Although they are well-known and standard algorithms, we will present them in the appendix [4] for completeness.

The search process is described in Algorithm 2. The key idea is that the sample closest to the query point on the lowest level will be its NN. Hence, by repeating the same process for inserting objects in the database, we can retrieve the NN w.h.p. We first bound the number of questions asked by Algorithm 1 (w.h.p.), in Theorem 3. Having this result, the proof of Theorem 2 is then immediate.

**Theorem 3.** *Algorithm 1 succeeds with probability higher than* $1 - \frac{1}{n}$*, and it requires asking no more than* $O(nD^3 \log^2 n + D \log^2 n \log \log n^{D^3})$ *questions w.h.p.*

We first state a technical lemma that we will need to prove Theorem 3. The proof could be found in Appendix [4].

**Lemma 2.** *Take $a$ a constant and* $\lambda_i = \frac{n}{(2D)^{i=1}}$*. For every object $p \in \mathcal{T} \cup \{q\}$, where $q$ is the query point, the following four properties of the data structure are true w.h.p.*

1. $|S_i \cap \beta_p(\lambda_{i+1})| \ge 1$    2. $|S_i \cap \beta_p(\lambda_i)| \le 4aD \log n$
3. $|S_{i+1} \cap \beta_p(\lambda_{i-1})| \le 16aD^3 \log n$    4. $|S_i \cap \beta_p(4\lambda_i)| \ge 4aD \log n$
5. $|S_{i+1} \cap \beta_p(4\lambda_{i-1})| \le 64aD^3 \log n$

*Proof of Theorem 3.* Let $m_i = a(2D)^i \log n$ denote the number of objects we sample at level $i$, and let $S_i$ be the set of samples at level $i$ *i.e.*, $|S_i| = m_i$. Here, $a$ is an appropriately chosen constant, independent of $D$ and $n$. Further, let $\lambda_i = \frac{n}{(2D)^{i-1}}$.

From now on, we assume that we are in the situation where Properties (1) to (5) in Lemma 2 are true for all objects (which is the case w.h.p.). Again, fix an object $p$. For each object $p$, we need to find $\nu_p(i)$, which is the nearest neighbor in $S_i$ with respect to $p$. In order for being able to continue this procedure in every level, we keep a wider range of objects: those objects in $S_i$ that have ranks less than $\lambda_{i+1}$ with respect to $p$ in level $i$; we store them in $\Gamma_p(i)$ (property 1 tells us that such objects exist), in this way the first object in $\Gamma_p(i)$ would be $\nu_p(i)$. In practice our algorithm stores some redundant objects in $\Gamma_p(i)$, but we claim that totally no more than $4aD \log n$ objects are stored in $\Gamma_p(i + 1)$. To summarize, the properties we want to maintain in each level are: 1- $\forall p \in \mathcal{T}$ and $1 \le i \le \log n / \log 2D$, $S_i \cap \beta_p(\lambda_i) \subseteq \Gamma_p(i)$ and 2- $|\Gamma_p(i)| \le 4aD \log n$.

<div style="border:1px solid">

**input** : A database with $n$ objects $p_1, ..., p_n$, and disorder constant $D$

**output**: For each object $u$, a vector $\nu_u$ of length $\log n / \log(2D)$. The list of all samples $\cup_i S_i$

**Def.:** $S_i$: The set of $a(2D)^i \log n$ random samples at level $i$, $i = 1, \ldots, \log n / \log(2D)$;

$\quad\quad \nu_o$: $\nu_o(i) =$nearest neighbor to object $o$ in $S_i$; $o \in \mathcal{T}$, $i = 1, \ldots, \log n / \log(2D)$;

$\quad\quad \Gamma_o(i)$: contains the $\lambda_i$ closest objects to $p$ in $S_i$, possibly with redundant objects;

$\quad\quad \Lambda_o(i)$: The set of $p \in S_i$, for which $\nu_p(i-1) \in \Gamma_o(i-1)$;

**for** $i \leftarrow 1$ **to** $L = \frac{\log n}{\log 2D}$ **do**
  **for** $p \leftarrow 1$ **to** $n$ **do**
    **if** $i = 1$ **then**
      | $\Lambda_p(1) \leftarrow S_1$
    **else**
      $\Lambda_p(i) = \{o \in S_i | \nu_o(i-1) \in \Gamma_p(i-1)\}$;
      **if** $|\Lambda_p(i)| = 0$ **then**
      | Report Failure
      **else**
        $H \leftarrow$ BuildHeap($\Lambda_p(i)$) ;
        **for** $k \leftarrow 1$ **to** $4aD \log n$ **do**
          $m \leftarrow$ ExtractMin($H$) ;
          add $m$ to $\Gamma_p(i)$
        **end**
      **end**
    $\nu_p(i) \leftarrow$ first object in $\Gamma_p(i)$;
  **end**
**end**

</div>

**Algorithm 1**: Learning Algorithm

<div style="border:1px solid">

**input** : A database with $n$ objects and disorder $D$, the list of samples, the vectors $\nu_u$ for $u \in \mathcal{T}$, a query point $q$

**output**: The nearest neighbor of $q$ in the database

$\Gamma_q(1) = S_1$;
**for** $i \leftarrow 2$ **to** $L = \frac{\log n}{\log 2D}$ **do**
  $\Lambda_q(i) \leftarrow \{p \in S_i | \nu_p(i-1) \in \Gamma_q(i-1)\}$;
  $H \leftarrow$ BuildHeap($\Lambda_q(i)$) ;
  **for** $k \leftarrow 1$ **to** $4aD \log n$ **do**
    $m \leftarrow$ ExtractMin($H$) ;
    add $m$ to $\Gamma_q(i)$
  **end**
**end**
**return** first object in $\Gamma_q(\frac{\log n}{\log 2D})$

</div>

**Algorithm 2**: Search Algorithm

In the first step, for all $p$, $\Lambda_p(1) = S_1$, and since $|S_1| = 2aD \log n < 4aD \log n$, all the objects in $S_1$ are extracted from the heap and therefore $\Gamma_p(i)$ is $S_1$ ordered with respect to $p$, as a result both the properties hold when $i = 1$. The argument for the maintenance of this property is as follows: Assume the property holds up to level $i$; we analyze level $i + 1$. In fact we want an object $s \in S_{i+1}$ to be in $\Gamma_p(i+1)$ if $r_p(s) \leq \lambda_{i+1}$ (note that Property 1 guarantees that there is a least one such sample). Further, let $s' \in S_i$ be the sample at level $i$ closest to $s$ *i.e.*, $s' = \min_{x \in S_i} r_s(s')$. Again, by Property 1, we know that $r_s(s') \leq \lambda_{i+1}$. Hence, by the approximate triangle inequality 3 (see Section 2), we have:

$$r_p(s, \mathcal{T}) \leq \lambda_{i+1} \text{ and } r_s(s', \mathcal{T}) \leq \lambda_{i+1} \quad \Rightarrow r_p(s', \mathcal{T}) \leq 2D\lambda_{i+1} = \lambda_i$$

hence $s' = \nu_s(i) \in S_i \cap \beta_p(\lambda_i) \subseteq \Gamma_p(i)$ using the first property for step $i$. Therefore $\nu_s(i) \in \Gamma_p(i)$ and therefore $s \in \Lambda_p(i+1)$. Property 2 tells us that $|S_{i+1} \cap \beta_p(\lambda_{i+1})| \leq 4aD \log n$. Hence by

taking the first $4aD \log n$ closest objects to $p$ in $\Lambda_p(i+1)$ and storing them in $\Gamma_p(i+1)$, we can make sure than both $s \in \Gamma_p(i+1)$ for $s \in S_{i+1}, s \in \beta_p(\lambda_{i+1})$ and $|\Gamma_p(i+1)| \leq 4aD \log n$.

Note that in the last loop of the algorithm when $i = \log n / \log 2D$, according to Property 1, $|S_i \cap \beta_p(\lambda_{i+1})| \geq 1$. But $\lambda_{i+1}$ in the last step is 1, therefore the closest object to $p$ in the database is in $S_{\log n / \log 2D}$, which means that $\nu_p(\log n / \log 2D)$ is the nearest neighbor of $p$ in the database. Repeating this argument for the query point in the Search algorithm shows that after the termination, the algorithm finds the nearest neighbor.

To analyze the complexity of the algorithm, we should show that $|\Lambda_p(i+1)|$ is not big. Property 4 tells us that all of the $4aD \log n$ closest samples to $p$ at level $i$ have rank less than $8\lambda_i$, so all objects in $\Lambda_p(i)$ have ranks less than $8\lambda_i$ with respect to $p$. Consider a sample $s \in S_i$ such that $r_p(s, \mathcal{T}) \leq 8\lambda_i$ and a sample $s'' \in S_{i+1}$ that falls in the bin of $s$.

If an object $s''$ is in $\Lambda_p(i+1)$, it means that it falls in the bin of an object $s$ in $\Gamma_p(i)$, i.e. $\nu_{s''}(i) \in \Gamma_p(i)$. Since $s \in \Gamma_p(i)$, we have $r_p(s, T) \leq 8\lambda_i$.

By property 1, we must have $r_{s''}(s, \mathcal{T}) \leq \lambda_{i+1}$. Thus, by inequality 2, we have:

$$r_{s''}(s, \mathcal{T}) \leq \lambda_{i+1} \text{ and } r_p(s, \mathcal{T}) \leq 8\lambda_i \Rightarrow r_p(s'', \mathcal{T}) < D(8\lambda_i + \lambda_{i+1}) \leq 4\lambda_{i-1}$$

By property 5, there are at most $O(D^3 \log n)$ such samples at level $i + 1$, i.e. $\Lambda_p(i+1) = O(D^3 \log n)$.

To summarize, at each level for each object, we build a heap out of $O(D^3 \log n)$ objects and apply $O(aD \log n)$ `ExtractMin` procedures to find the first $4aD \log n$ objects in the heap. Each `ExtractMin` requires $O(\log(D^3 \log n)) = O(\log \log n^{D^3})$. Hence the complexity for each level and for each object is $O(D^3 \log n + D \log n \log \log n^{D^3})$. There are $O(\log n)$ levels and $n$ objects, so the overall complexity is $O(nD^3 \log n + nD \log^2 n \log \log n^{D^3})$. $\qquad\square$

*Proof of Theorem 2.* The upper bound on the number of questions to be asked in the learning phase is immediate from Theorem 3. For each object, we need to store one identifier (the identifier of the closest object) at every level $i$ in the hierarchy, and one bit to mark it as a member of $S_i$ or not; also one bit if it is in $\Gamma_q(i-1)$ and one bit for being in $\Lambda_q(i)$ (we can reuse this memory in the next level) (note that a heap with size N needs $O(N \log n)$ memory, where $\log n$ is for storing each object). Hence, the total memory requirement[2] do not exceed $O(n \log^2 n / \log(2D))$ bits. Finally, the properties 1-5 shown in the proof of Theorem 3 are also true for an external query object $q$. Hence, to find the closest object to $q$ on every level, we build the same heap structure, the only difference is that instead of repeating this procedure $n$ times in each level, since there is just one query point, we need to ask at most $O(D^3 \log^2 n + D \log^2 n \log \log n^{D^3})$ questions totally. In particular, the closest object at level $L = \log_{2D}(n)$ will be $q$'s nearest neighbor w.h.p. $\qquad\square$

Note that this scheme can be easily modified for $R$-nearest neighbor search. At the $i$-the level of the hierarchy, the closest sample to $q$ will, w.h.p., be one of its $\frac{n}{(2D)^i}$ nearest neighbors. If we are only interested in the level of precision, we can stop the hierarchy construction at the desired level.

# 6   Discussion

The use of a comparison oracle is motivated by a human user who can make comparisons between objects but not assign meaningful numerical values to similarities between objects. There are many interesting questions raised by studying such a model including fundamental characterizations of the complexity of search in terms of number of oracle questions. We also believe that ideas of searching through comparisons form a bridge between many well known search techniques in metric spaces to perceptually important (non-metric spaces) situations, and could lead to innovative practical applications. Analogous to locality sensitive hashing, one can develop notions of rank-sensitive hashing, where "similar" objects based on ranks are given the same hash value. Some preliminary ideas for it were given in [3], but we believe this is an interesting line of inquiry. Also in [3], we have implemented comparison-based search heuristics to navigate image database.

## Footnotes

[1]in fact the value of $a$ is dependent on the value of error we expect, the more accurate we want to be, the more sample points we need in each level and $a$ would be larger.

[2] Making the assumption that every object can be uniquely identified with $\log n$ bits

# References

[1] N. Goyal, Y. Lifshits, and H. Schutze, "Disorder inequality: A combinatorial approach to nearest neighbor search," in *WSDM*, 2008, pp. 25–32.

[2] S. Dasgupta and Y. Freund, "Random projection trees and low dimensional manifolds," in *STOC*, 2008, pp. 537–546.

[3] D. Tschopp, "Routing and search on large scale networks," Ph.D. dissertation, École Polytechnique Fédérale de Lausanne (EPFL), 2010.

[4] D. Tschopp, S. Diggavi, P. Delgosha, and S. Mohajer, "Randomized algorithms for comparison-based search: Supplementary material," 2011, submitted to NIPS as supplementary material.

[5] K. Clarkson, "Nearest-neighbor searching and metric space dimensions," in *Nearest-Neighbor Methods for Learning and Vision: Theory and Practice*, G. Shakhnarovich, T. Darrell, and P. Indyk, Eds. MIT Press, 2006, pp. 15–59.

[6] Y. Rubner, C. Tomasi, and L. J. Guibas, "The earth mover's distance as a metric for image retrieval," *International Journal of Computer Vision*, vol. 40, no. 2, pp. 99–121, 2000.

[7] E. Demidenko, "Kolmogorov-smirnov test for image comparison," in *Computational Science and Its Applications - ICCSA*, 2004, pp. 933–939.

[8] M. Nachtegael, S. Schulte, V. De Witte, T. Mlange, and E. Kerre, "Image similarity, from fuzzy sets to color image applications," in *Advances in Visual Information Systems*, 2007, pp. 26–37.

[9] S. Santini and R. Jain, "Similarity measures," *IEEE transactions on Pattern Analysis and Machine Intelligence*, vol. 21, no. 9, pp. 871–883, 1999.

[10] G. Chechik, V. Sharma, U. Shalit, and S. Bengio, "Large scale online learning of image similarity through ranking," *Journal of Machine Learning Research*, vol. 11, pp. 1109–1135, 2010.

[11] A. Frome, Y. Singer, F. Sha, and J. Malik, "Learning globally-consistent local distance functions for shape-based image retrieval and classification," in *ICCV*, 2007, pp. 1–8.

[12] Y. Lifshits and S. Zhang, "Combinatorial algorithms for nearest neighbors, near-duplicates and small-world design," in *SODA*, 2009, pp. 318–326.

[13] R. Krauthgamer and J. R. Lee, "Navigating nets: simple algorithms for proximity search," in *SODA*, 2004, pp. 798–807.

[14] D. R. Karger and M. Ruhl, "Finding nearest neighbors in growth-restricted metrics," in *STOC*, 2002, pp. 741–750.

[15] T. Cormen, C. Leiserson, R. Rivest, and C. Stein, "Introduction to algorithms," *MIT Press and McGraw-Hill Book Company*, vol. 7, pp. 1162–1171, 1976.

[16] R. Motwani and P. Raghavan, *Randomized Algorithms*. Cambridge University Press, 1995.

